# An Unsupervised Decontamination Procedure For Improving The Reliability Of Human Judgments

**Michael C. Mozer,**[*] **Benjamin Link,**[*] **Harold Pashler**[†]
[*] Dept. of Computer Science, University of Colorado
[†]Dept. of Psychology, UCSD

## Abstract

Psychologists have long been struck by individuals' limitations in expressing their internal sensations, impressions, and evaluations via rating scales. Instead of using an absolute scale, individuals rely on reference points from recent experience. This *relativity of judgment* limits the informativeness of responses on surveys, questionnaires, and evaluation forms. Fortunately, the cognitive processes that map stimuli to responses are not simply noisy, but rather are influenced by recent experience in a lawful manner. We explore techniques to remove sequential dependencies, and thereby *decontaminate* a series of ratings to obtain more meaningful human judgments. In our formulation, the problem is to infer latent (subjective) impressions from a sequence of stimulus labels (e.g., movie names) and responses. We describe an unsupervised approach that simultaneously recovers the impressions and parameters of a contamination model that predicts how recent judgments affect the current response. We test our *iterated impression inference*, or $\mathbb{I}^3$, algorithm in three domains: rating the gap between dots, the desirability of a movie based on an advertisement, and the morality of an action. We demonstrate significant objective improvements in the quality of the recovered impressions.

## 1 Introduction

Individuals are often asked to convey their opinions and sentiments in the form of quantitative judgments. On a 1–5 scale, how much did you enjoy the movie Kung Fu Panda? How many stars would you give the Olive Garden restaurant? How bad is the pain in your back, where 1 means no pain and 10 means unbearable? What grade should you assign to the term paper on "Consciousness and Commander Data"? On a Likert scale (ranging from strongly disagree to strongly agree), what is your attitude toward the statement "NIPS should stay in North America"?

Researchers in the social sciences have developed methods for minimizing response bias of various sorts (Bagozzi, 1994). Response bias can occur from the wording of questions, respondents trying to portray themselves in a certain way, individual differences in the use of the response scale (e.g., extreme responding versus midpoint responding), or even cultural variation in the ideal rating-scale granularity (Chami-Castaldi, Reynolds, & Wallace, 2008). An additional influence on responses is the sequential ordering of items to be judged. To illustrate, suppose you are asked to make a series of moral judgments concerning various actions using a 1-10 scale, with a rating of 1 indicating 'not particularly bad or wrong' and a rating of 10 indicating 'extremely evil.' When individuals are shown the series on the left, their ratings of item (3) tend to be higher than the identical item (3′) in the series on the right (Parducci, 1968).

| | |
|---|---|
| (1) Stealing a towel from a hotel | (1′) Testifying falsely for pay |
| (2) Keeping a dime you find on the ground | (2′) Using guns on striking workers |
| (3) Poisoning a barking dog | (3′) Poisoning a barking dog |

Individuals seem incapable of making absolute judgments, and instead recent experience provides reference points with respect to which relative judgments are made (e.g., Laming, 1984; Parducci,

1965, 1968; Stewart, Brown, & Chater, 2005). These *sequential dependencies* in judgment arise in almost every task in which an individual is asked to make a series of responses, such as filling out surveys, questionnaires, and evaluations (e.g., usability ratings, pain assessment inventories). Every faculty member is aware of drift in grading that necessitates comparing papers graded early on a stack with those graded later. Sequential effects have been demonstrated in domains as varied as legal reasoning and jury evidence interpretation (Furnham, 1986; Hogarth & Einhorn, 1992) and clinical assessments (Mumma & Wilson, 2006).

Sequential dependencies are observed in even the simplest of laboratory tasks involving the rating of unidimensional stimuli, such as the loudness of a tone or the length of a line. Individuals' ability to rate even these simple stimuli is surprisingly poor compared to their ability to discriminate the same stimuli. Across many domains, responses convey not much more than two bits of mutual information with the stimulus (Stewart et al., 2005). The poor transmission of information is attributed in large part to the contamination from recent trials. (A *trial* is psychological jargon for a single judgment of a stimulus.)

We (Mozer et al., 2010) have surveyed the empirical and theoretical literature on sequential effects (e.g., DeCarlo & Cross, 1990; Parducci, 1965; Petrov & Anderson, 2005; Stewart et al., 2005), and mention here findings relevant for understanding their mechanistic basis. The influence of recent trials is exerted by *both* stimuli and responses. We'll refer to the relevant recent history as the *context*. One interpretation of the joint effect of stimuli and responses is that individuals form their current response by analogy to recent trials: they determine a response to the current stimulus that has the same relationship as the previous response had to the previous stimulus. The immediately preceding trial has the strongest influence on responses, and the influence of further back trials typically falls off exponentially. Linear autoregression models have done a reasonable job of accounting for sequential dependencies, though many theories include nonlinearities, e.g., memory based anchors, and generalization from previous trials that depends on the similarity of the current stimulus to the previous stimuli.

## 2 Decontamination Models

Because responses are influenced by recent context, they are not as pure a measure of an individual's stimulus appraisal as one might wish for. In the applied psychology literature, techniques have been explored to mitigate judgment relativity effects, such as increasing the number of response categories and varying the type and frequency of anchors (Mumma & Wilson, 2006; Wedell, Parducci, & Lane, 1990).

In previous work (Mozer et al., 2010), we proposed an alternative: an algorithmic technique that *decontaminates* responses to remove contextual influences. By removing the contamination from previous trials, we recover ratings that are more meaningful than are the raw ratings. We focused on a task in which an objective ground truth exists in order that we could assess the quality of the ratings. The task involved judging the gap between two dots that appear on a monitor. In our experiments, ten equally spaced gaps were used, and we asked participants to rate the gaps on a 1-10 scale. Even though there is a one-to-one mapping between stimuli and responses, and participants were shown all gaps prior to the start of the experiment, participants still show strong sequential dependencies in their ratings. Our decontamination procedure obtains ratings that better reflect ground truth than the reported ratings by about 5%. (This improvement is purely due to desequencing—the removal of sequential effects. The improvement rises to about 20% with debiasing and decompressing the ratings.)

Our framework assumes that an external stimulus—the dot pairs in our experiment—maps to an internal mental representation we refer to as the *impression*, and this impression is then mapped to a response. We treat the mapping from stimulus to impression as veridical, and contamination from recent trials as occurring in the mapping from impression to response. The term *sensation* might be preferred over impression if the judgment task is purely perceptual, and the term *evaluation* might be preferred in a domain involving higher cognition, but we will use impression as the generic term for the internal representation. The goal of decontamination is to recover the (latent) impression from the sequence of ratings and stimuli. To introduce some notation, $S_t$ denotes the stimulus presented on trial $t$, $\alpha_{S_t}$ denotes the impression associated with the stimulus, and the corresponding rating or response is $R_t$. $S_t$ is a unique label associated with each distinct stimulus. Importantly, we do

not assume to have any metric or featural information about the stimulus; we will simply index the $n$ distinct stimuli with integers, $1, 2, ..., n$. We denote the impression associated with stimulus $s$ as $\alpha_s$. Decontamination involves discovering $\boldsymbol{\alpha} \equiv \{\alpha_1, \alpha_2, ..., \alpha_n\}$ given stimulus sequence $\{S_1, S_2, ..., S_T\}$ and the corresponding response sequence $\{R_1, R_2, ..., R_T\}$, where each of the $n$ distinct stimuli is presented at least once in the stimulus sequence, i.e., $\forall s \in \{1, 2, ..., n\}, \exists t : S_t = s$.

In our previous work, we utilized ground truth not only to evaluate the quality of decontamination procedures but also for training decontamination models. That is, we adopted a supervised training paradigm in which the ground truth provided the target impression values. We built a single model for all participants. One group of participants was used for training the model, and another group for testing. We explored a range of models, including linear and nonlinear regression, look up tables, hybrid models, and these same models embedded in a conditional random fields (CRFs). With their more powerful inference techniques, the CRF-based models performed the best.

Ground truth is known for stimuli that vary along a unidimensional perceptual continuum, e.g., gaps between points, pitches of tones. However, interesting and realistic judgment tasks often involve stimuli that vary along dimensions that are ill defined and even inherently subjective (i.e., a universal ground truth does not exist) Even perceptual tasks may have this character, e.g., smell or taste evaluation. And in cognitive preference tasks, e.g., the rating of movies or music, not only are the stimulus dimensions unknown but the critical dimensions and preferences may vary from one individual to the next.

In complex, cognitive domains, the only means of obtaining ground truth for an individual is to ask the individual to rate the same stimulus in many contexts and to average the ratings to eliminate the "noise" due to sequential effects. Although training data can in principle be obtained, the cost is nontrivial. In our simple gap-measurement task, even with *twenty* ratings of the each gap, the error in the impression obtained by debiasing, decompressing, and averaging ratings was still nonzero, and dropped with each subsequent rating incorporated into the average.

## 2.1 The Iterated Impression Inference ($\mathbb{I}^3$) Algorithm

Given the challenge of collecting sufficient ground-truth data for supervised training of decontamination models, our goal in this paper is to develop an *unsupervised* technique for decontamination of rating sequences. Our technique involves simultaneously inferring the set of impressions, $\boldsymbol{\alpha}$, and the parameters $\boldsymbol{\beta}$ of a contamination model $\mathcal{C}_{\boldsymbol{\beta}}$, that predicts the response at time $t$ given the current stimulus and context:

$$\hat{R}_t = \mathcal{C}_{\boldsymbol{\beta}} \left( \alpha_{S_t}, ..., \alpha_{S_{t-h}}, R_{t-1}, ..., R_{t-h} \right),$$

where $\hat{R}_t$ denotes the prediction and $h$ is the number of trials of history (the context) used to make the prediction. In the style of the EM algorithm, our approach is a straightforward iteration between inference on the latent variables and updating the model parameters:

1. Define the baseline estimate of the impression for each stimulus $s$ to be the average rating given on all trials when $s$ is presented, i.e.,

$$\alpha_s^0 = E_{\{t:S_t=s\}} \left[ R_t \right],$$

   where the superscript (0) associated with $\alpha$ indicates the iteration of the algorithm, and $E$ is the expectation over a set of trial indices.

2. Given impressions determined on the previous iteration $i$, $\boldsymbol{\alpha}^i$, train a new contamination model for iteration $i+1$, $\mathcal{C}_{\boldsymbol{\beta}^{i+1}}$, by searching for model parameters $\boldsymbol{\beta}^{i+1}$ that minimize the mean squared-error, $\text{MSE}(\boldsymbol{\alpha}^i, \boldsymbol{\beta}^{i+1})$, defined as

$$\text{MSE}(\boldsymbol{\alpha}, \boldsymbol{\beta}) = E_t \left[ \left( \mathcal{C}_{\boldsymbol{\beta}} \left( \alpha_{S_t}, ..., \alpha_{S_{t-h}}, R_{t-1}, ..., R_{t-h} \right) - R_t \right)^2 \right].$$

3. Given the updated contamination model, $\mathcal{C}_{\boldsymbol{\beta}^{i+1}}$, search for a new set of impressions, $\boldsymbol{\alpha}^{i+1}$, that minimize the mean squared-error criterion $\text{MSE}(\boldsymbol{\alpha}^{i+1}, \boldsymbol{\beta}^{i+1})$.

4. Repeat steps 2 and 3 until $\boldsymbol{\alpha}^{i+1} \approx \boldsymbol{\alpha}^i$.

We refer to this algorithm as *Iterated Impression Inference* or $\mathbb{I}^3$. Because the MSE is strictly non-increasing at steps 2 and 3, $\mathbb{I}^3$ is guaranteed to find a local optimum in the search space. The initialization at step 1 starts the search in the neighborhood of the solution because—by our definition of an impression—the responses are noise-corrupted and recency-modulated instantiations of the impressions. Consequently, local optimization has a shot at finding a good solution.

The simulations described in the following sections all use a simple linear model for $\mathcal{C}_\beta$,

$$\hat{R}_t = \beta_0 + \beta_1 \alpha_{S_t} + ... + \beta_{h+1} \alpha_{S_{t-h}} + \beta_{h+2} R_{t-1} + ... + \beta_{2h+1} R_{t-h}.$$

Because the model is bilinear in the set of variables that we're solving for— $\boldsymbol{\alpha}$ and $\boldsymbol{\beta}$—steps 2 and 3 each amount to solving a least squares regression problem, and the $\mathbb{I}^3$ algorithm offers an approach to bilinear regression. Although this system of bilinear equations does not have a unique least-squares solution, the $\mathbb{I}^3$ benefits from the strong initialization conditions. Related iterative algorithms for bilinear regression are found in the literature (Bai & Li, 2004; Bai & Liu, 2006).

In preference tasks (e.g., movie rating) where impressions are different for different individuals, $\mathbb{I}^3$ has a large number of free parameters: an impression $\alpha$ must be inferred for each distinct item being rated *and* for each respondent. To address the possibility of overfitting, we incorporate ridge regression in estimating the impressions (step 3 of $\mathbb{I}^3$). As the regressand at step 3, we use the deviation of the impression at iteration $i$ from the baseline impression, i.e., $\boldsymbol{\alpha}^i - \boldsymbol{\alpha}^0$. Consequently, regularization penalizes large deviations from the baseline impressions, and a large ridge parameter prevents the impressions from wandering too far from the baseline. Overfitting is avoided because the baseline impressions are grounded in the ratings.

## 3 Simulations

We describe a series of simulations using $\mathbb{I}^3$ to decontaminate both artificial sequences and actual rating sequences from behavioral experiments. In all cases, ratings are integers on a 1–10 scale. The impressions are on the same scale, but are allowed to be continuous in $[1, 10]$. With $||\boldsymbol{\beta}|| = 2h + 2$, $||\boldsymbol{\alpha}|| = n$, and a context of $h$ trials required before the model can be used, we need a $T \geq 3h + n + 2$ trial sequence to constrain the model from the data. In all our experiments, two complete passes through the (randomly ordered) set of stimuli is sufficient to constrain the model parameters, although we could in principle get by with fewer than two complete passes through the stimuli.

Data from multiple participants are obtained in each experiment. In principle, we could decontaminate each participant's data in isolation. However, in the simulations we report, we have chosen to build a single contamination model for all participants, thereby imposing a strong constraint on the $\boldsymbol{\beta}$ parameters. An alternative would be a hierarchical Bayesian approach with shared hyperpriors but separate parameter inference for each participant.

### 3.1 Artificial Data

To evaluate $\mathbb{I}^3$ under ideal circumstances, we construct artificial data via a generative contamination process that is consistent with the linear form of $\mathcal{C}$, and therefore $\mathbb{I}^3$ should be able to perform perfect decontamination given sufficient data. The artificial sequences are generated by drawing randomly from $n = 10$ stimuli for a total of $p$ passes through the stimulus set such that each stimulus appears exactly once in a series of 10 trials and the total number of trials is $T = pn$. The impressions associated with each stimulus were randomly drawn from $\{2, 4, 6, 8\}$, and responses were generated by an autoregressive model: $R_t = \alpha_{S_t} + \alpha_{S_{t-1}} - R_{t-1}$, anchored with $\alpha_{S_0} - R_0 \equiv -1$. For example, the impression sequence $\{8, 2, 4, 8, 6, 4\}$ would yield response sequence $\{7, 3, 3, 9, 5, 5\}$. The stimulus-impression mapping was used in the generative process, but was not provided to $\mathbb{I}^3$. The goal of $\mathbb{I}^3$ is to infer both the impressions $\boldsymbol{\alpha}$ and the model parameters $\boldsymbol{\beta}$.

Figure 1 shows the results of 50 replications of the simulation in which we vary the number of participants, from 1 to 5, and the number of passes, $p$, from 2 to 10. For each replication, we compute the mean squared-error (MSE) between the true impressions and $\boldsymbol{\alpha}^0$—the baseline recovered impression that is obtained by averaging ratings across stimulus presentations. We also compute the MSE between the true impressions and $\boldsymbol{\alpha}^\infty$—the impressions recovered by $\mathbb{I}^3$. The percentage improvement due to $\mathbb{I}^3$ is displayed on the ordinate of the graph. Even with only one participant and $T = 20$ trials, $\mathbb{I}^3$ reduces the error in the reconstructed impression by 65%. With 3 or more

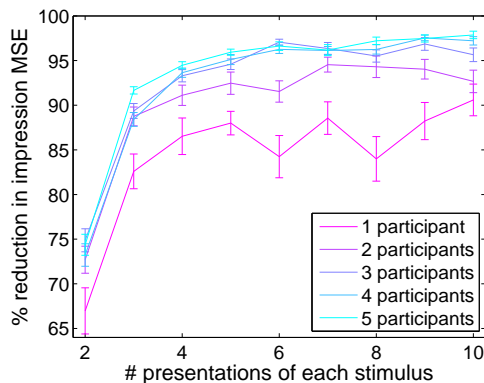

Figure 1: Benefit of decontamination (% reduction in MSE for decontamination over baseline) for the artificial data set as a function of number of stimulus presentations and number of individuals providing data. Error bars are $\pm 1$ SEM.

participants, and $T = 50$ trials, the error is reduced by 95%. To emphasize, this reduction is due to the use of decontamination and reflects the improvement over a baseline which is the best one can do without considering sequential dependencies, treating variability in responses to the same stimulus simply as noise to be eliminated by averaging.

Comparison of the curves for, say, one versus two participants in Figure 1 indicates a benefit of combining data to build a single contamination model for all participants, assuming, of course, that the participants share a common underlying contamination process.

## 3.2 Gap-Estimation Task

Next, we decontaminate data from the gap-estimation task described earlier (Mozer et al., 2010). The reason for using this task is that it provides ground truth, which can be used for evaluating the quality of the recovered impressions even though $\mathbb{I}^3$ does not use ground truth impressions in training. The experiment consists of 180 trials in which pairs of dots were presented and participants were asked to estimate the gap between dots. In every block of 10 consecutive trials, the set of distinct gaps were shown in random order. Data were collected from 76 participants.

Because the trials are blocked, we can vary the number of passes, $p$, through the stimulus set used for decontamination by selecting only the first $T = 10p$ trials of the experiment. Figure 2a shows the MSE associated with $\alpha^0$, the baseline impression, and $\alpha^\infty$, the impression recovered by $\mathbb{I}^3$, as a function of $p$. The key feature of the curves is that increasing $p$—obtaining more ratings of each stimulus—produces a steep drop in MSE. Surprisingly, even with 18 ratings of the same stimulus, there is still a significant discrepancy between ground truth and the mean rating provided by the participant. This result is all the more surprising considering that we postprocess the recovered impressions to use the full response range; without this postprocessing, the error is larger yet.

Although $\mathbb{I}^3$ produces impressions closer to ground truth than the baseline for $p$ between 2 and 12, Figure 2a belies the magnitude of the difference. The solid green curve of Figure 2b shows the percentage reduction in MSE due to $\mathbb{I}^3$. When only a small number of ratings of each stimulus is available, $\mathbb{I}^3$ obtains a roughly 10% reduction in MSE by accounting for sequential dependencies. We can evaluate the significance of this reduction by asking what percentage reduction in MSE would be obtained if we simply collected $p + 1$ ratings of each stimulus instead of $p$. The benefit of additional data is shown in the dashed blue curve of Figure 2b. For small $p$, collecting additional data is more valuable than decontaminating the existing data, but once we have roughly $p = 4$ samples, the benefit of decontamination is comparable to the benefit obtained from collecting an additional round of ratings. However, the cost of collecting additional ratings can be quite high if you consider large data sets, e.g., music and movies.

Figure 2c depicts the distribution of errors for individual participants and stimuli, where the red points indicate the (signed) error in the baseline impression for $p = 3$, with one point per participant and stimulus, and the green points indicate the error in the impressions recovered by $\mathbb{I}^3$ for $p = 3$. We have jittered the horizontal position of the points a bit to make it easier to see the densities. It's clear that the large errors are reduced by decontamination, and the other points appear more tightly clustered around zero. (The errors for the endpoints of the stimulus continuum are small because of the rescaling we mentioned previously.)

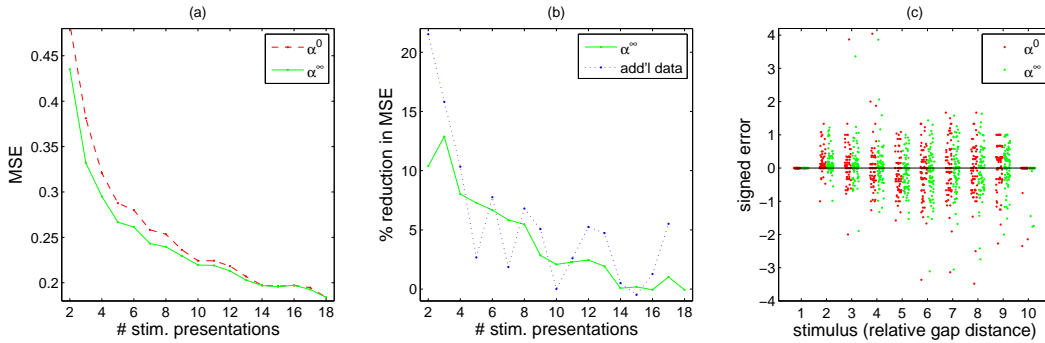

Figure 2: (a) Mean squared-error of the baseline impressions ($\alpha^0$) and impressions recovered by $\mathbb{I}^3$ ($\alpha^\infty$) on the gap-estimation task as a function of number of stimulus presentations included in the modeling. (b) Percentage reduction in MSE for $\mathbb{I}^3$ over baseline (green line); benefit of adding an additional stimulus presentation to the data set (dashed blue). (c) Distribution of errors for individual participants and stimuli

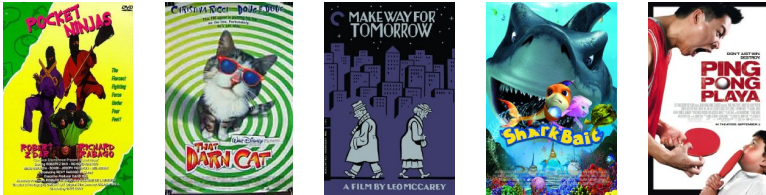

Figure 3: Movie ads used for rating task. Examples are shown of action, comedy, drama, family, and sports genres.

### 3.3 Movie Advertisement Evaluation

Having established the value of decontamination in a domain where performance can be assessed relative to objective truth, we move on to examine judgments in more complex, subjective domains. We conducted a web-based experiment in which participants were asked to indicate their desire to see a movie based solely on a movie poster of the sort that typically appears on a DVD jacket, and shows images from the movie, the movie title, and sometimes quotes from reviews (Figure 3). Participants were asked to rate each movie on a 1–10 scale, where 1 means "would never watch this movie" and 10 means "can't wait to see it". The rating task here should not be confused with more typical rating task of indicating enjoyment for a previously viewed film; this sort of task might be used by film marketers who attempt to design advertisements to have broad appeal.

We selected 50 relatively obscure movies from the Internet Movie Database (IMDb.com). Obscurity was determined by a small number of user ratings on IMDb. We polled participants during the experiment to verify that the films were generally unfamiliar. We chose 10 movies from each of five genres: action, comedy, drama, family, and sports. The movies varied in their mean IMDb rating and in their release year, from 1947 to 2007. Participants were asked to rate each movie four times for a total of $T = 200$ trials. The trials were blocked such that each movie was presented exactly once every 50 trials. The movies within a block were ordered randomly with the constraint that consecutive films were always drawn from different genres. We collected data from 120 participants in the United States using Mechanical Turk, rejecting five whose ratings looked suspicious on several criteria. Ordinarily, tasks for Mechanical Turk workers are defined to be a single trial; we set up a javascript sequence of 200 trials that had to be completed by the worker to receive payment. We required that participants respond on each trial within 15 seconds to ensure a steady rate of responding.

Because of the large stimulus set in this experiment—in contrast to the previous experiment with just 10 items—we had sufficient data to perform model selection by cross validation. We split the data from each participant into 100 trials for training and 100 trials for testing. Within the 100 training trials, we used trials 1-80 for training $\mathbb{I}^3$ and trials 81-100 as a validation set for model selection. We searched over two hyperparameters previously described: $h$, the number of contextual trials to include in the model, and the ridge (regularization) parameter. We say more about $h$ shortly.

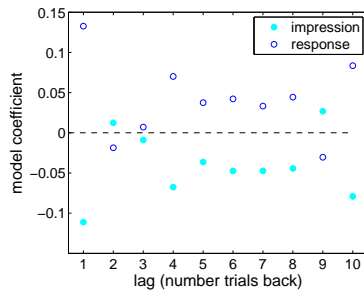

Figure 4: Regression model coefficients for impression (open circles) and response (closed circles) terms, for trial lags 1–10.

One way to evaluate the quality of the contamination model and impressions inferred by $\mathbb{I}^3$ is to use the model to predict ratings in the test set. Even though our goal is to decontaminate ratings, if we are successful at this goal we should be able to predict the consequence of context on ratings. We predicted ratings in the test set either with the baseline impressions, $\boldsymbol{\alpha}^0$, or with the combination of the contamination model and the recovered impressions, $\boldsymbol{\alpha}^\infty$. We obtained a 5.18% reduction in the test set prediction error with $\mathbb{I}^3$ over the baseline approach that does not exploit sequential dependencies. Of this reduction in error, about 2/3 (3.32%) was attributable to modeling sequential dependencies and 1/3 (1.86%) to iterative impression inference.

Having established the quality of the model using the test set, we can ask a more substantive question about how the model treats movie genre. Intuitively, one would expect individuals to cluster their ratings within genre. Some individuals will love dramas and hate comedies; others will have the reverse preferences. This clustering is present in the data. Using the baseline impressions $\boldsymbol{\alpha}^0$, we computed the ratio of intra- to inter-genre impression variance and compared it to a shuffled measure in which we randomly reassigned movies to genres. The original data has a variance ratio of 8.41, whereas the shuffled data has a ratio of 19.8, highly significant ($p < .0001$) by a sign test with participants as the random factor. A further indication of the quality of the inferred impressions, $\boldsymbol{\alpha}^\infty$, is that they obtain an even more compact clustering by genre: the variance ratio for the inferred impressions is 8.04, a reliable 4.40% reduction in the ratio ($p < .0005$ by a sign test).

The model we present here used $h = 10$ time steps of history, chosen by cross validation. We searched $h$ in the range of 1–10, but didn't go beyond 10 because each additional time step causes the loss of a training trial. We were surprised to find this amount of context showing a benefit, but we believe we have an explanation that suggests the model is capturing slow drift as well as very local influences of the sequence. Figure 4 shows the model coefficients for impression and response terms for lags 1–10. (The lag 0 impression coefficient is 1.0.) To a first order, the impression and response coefficients are symmetric with most of the impression coefficients being negative. If one thinks of the impression as the average response to a stimulus, then these weights will tend to lower the predicted response on the current trial if recent trials have produced stimulus-conditioned responses that are lower than the average stimulus-conditioned response, and vice-versa, i.e., slow drift. Beyond this first-order analysis of the weights, note that the impression and response coefficients are not entirely symmetric: the impression coefficients tend to have smaller magnitudes. Further, the overall magnitude is larger at lag 1, and the lag 1 and 2 coefficients have flipped signs, a typical pattern reflecting assimilation to the trial $t-1$ and contrast with trial $t-2$ (DeCarlo & Cross, 1990).

### 3.4 Morality Judgments

We conducted a final experiment in which participants were asked to rate the morality of various actions, like the Parducci experiment cited in the introduction of this paper. We concocted 25 actions, ranging from relatively inoffensive (*picking two lemons from your neighbors' tree without their permission*) to questionable (*failing to report $3000 in cash earnings on a tax return*) to the unimaginable (*sentencing a rape victim to death to prevent her from carrying a child to term*). The experiment consisted of $T = 100$ trials, in blocks of 25 trials in which each action was presented exactly once. Fifty participants were enlisted using Mechanical Turk, using the same procedure for collecting a sequence of ratings as we used for the movie-ad experiment. Three additional participants were rejected due to suspicious patterns in their data (e.g., all items assigned the same rating). Seventy-five trials were used for training the model, and 25 for testing. Of the 75 training trials, 50

were used for setting model parameters and 25 were used for model selection on $h$ and the ridge parameter. In this data set, $h = 4$ yielded the best validation (rating prediction) performance.

As summarized in Table 1, we found a benefit for decontamination in these data, although perhaps the magnitudes are a bit smaller than in the movie-ad data. The reduction in error in the test set that comes about by predicting responses using $\mathbb{I}^3$—relative to using the baseline impressions and not accounting for sequential dependencies—is 4.5%. Beyond this basic verification that the inferred impressions are valid, we hypothesized that in the domain of moral judgments—in contrast to movie-ad sentiment—there should be a strong consensus among individuals within the same culture. Consequently, impressions gleaned from the ratings should show a high degree of interrater agreement. We can measure the interrater agreement as the ratio of the variance of ratings to an item over participants to the variance of mean ratings over items. By this measure, the impressions inferred by $\mathbb{I}^3$, $\alpha^\infty$, are superior to the baseline impressions, $\alpha^0$, in that the interrater agreement measure improves by 2.13%. Although this improvement is small, it is highly consistent across items ($p < .005$ by a sign test with items as the random factor).

In this experiment, there are only 25 items and many of them are quite distinctive and clearly at the ends of any continuum of actions. Consequently, participants are likely to remember not only having rated items previously, but also the ratings that they assigned. To the extent that memory is playing a role in this experiment, it will diminish sequential effects and the potential of a model like $\mathbb{I}^3$ to improve the quality of inferred impressions. It seems advisable in future work to use larger sets of items, or to impose a waiting period between passes through the items.

## 4 Discussion

In this paper, we posed the challenge of improving the quality of human judgments by partialing out contextual contamination. Although both the phenomenon and theory of sequential dependencies has been studied in the psychology literature for over half a century, our work is aimed at the more practical concern of mitigating the influence of recent trials, in order to remove an important source of uncontrolled variability in the data.

In this work, we've tried to assess the practical utility of decontamination. In the gap-measurement task, we showed that decontamination reduced mismatch between ratings and ground truth about as much as using an additional round of ratings to smooth out the average. With a large set of items to be rated, the time savings can be significant. In the movie ad and morality tasks, we showed a roughly 5% improvement in rating predictability with decontamination, a nontrivial improvement considering that the Netflix prize was aimed at obtaining a 10% improvement in total. Further, decontamination recovered impressions that were more sensible and therefore more meaningful, in the sense that the impressions were more consistent within genres for movie ads and were more consistent across respondents for morality judgments.

$\mathbb{I}^3$ can readily be incorporated into current web-based and pencil-and-paper surveys if respondents are asked to rate some items more than once. Rating the items twice is sufficient in the tasks we studied to show a benefit of decontamination, but in principle, the model parameters can be constrained with fewer than two complete passes through the items. Further, if a training pool of subjects is used to constrain model parameters (e.g., to set $\boldsymbol{\beta}$ or to establish priors on $\boldsymbol{\beta}$ and $\boldsymbol{\alpha}$), it's conceivable that decontamination will work without requiring much more than a single rating per item. This final point suggests an obvious avenue for further research: exploring more sophisticated, Bayesian approaches that can better exploit cross-participant constraints to improve the quality of decontamination or reduce the amount of data that needs to be collected to perform decontamination.

| Experiment | Reduction in test set prediction error | Improvement in clustering of genre ratings | Improvement in interrater reliability |
|---|---|---|---|
| Movie ads | 5.18% | 4.40% ($p < .0005$) | |
| Morality | 4.46% | | 2.13% ($p < .005$) |

Table 1: Summary of results from movie ad and morality experiments

**Acknowledgments**

This research was supported by NSF grants BCS-0339103 and BCS-720375.

**References**

Bagozzi, R. P. (1994). Measurement in marketing research: Basic principles of questionnaire design. In R. P. Bagozzi (Ed.), *Principles of marketing research.* Massachusetts, USA: Basil Blackwell Ltd.

Bai, E.-W., & Li, D. (2004). Convergence of the iterative Hammerstein system identification algorithm. *IEEE Transactions on Automatic Control*, *49*, 1929–1940.

Bai, E.-W., & Liu, Y. (2006). Least squares solutions of bilinear equations. *Systems and Control Letters*, *55*, 466–472.

Chami-Castaldi, E., Reynolds, N., & Wallace, J. (2008). Individualised rating-scale procedure: a means of reducing response style contamination in survey data? *Electronic Journal of Business Research Methods*, *6*, 9–20.

DeCarlo, L. T., & Cross, D. V. (1990). Sequential effects in magnitude scaling: Models and theory. *Journal of Experimental Psychology: General*, *119*, 375–396.

Furnham, A. (1986). The robustness of the recency effect: Studies using legal evidence. *Journal of General Psychology*, *113*, 351–357.

Hogarth, R. M., & Einhorn, H. J. (1992). Order effects in belief updating: The belief adjustment model. *Cognitive Psychology*, *24*, 1–55.

Laming, D. R. J. (1984). The relativity of "absolute" judgements. *Journal of Mathematical and Statistical Psychology*, *37*, 152–183.

Mozer, M. C., Pashler, H., Wilder, M., Lindsey, R., Jones, M. C., & Jones, M. N. (2010). Decontaminating human judgments to remove sequential dependencies. In J. Lafferty, C. K. I. Williams, J. Shawe-Taylor, R. S. Zemel, & A. Culota (Eds.), (p. 1705-1713). La Jolla, CA: NIPS Foundation.

Mumma, G. H., & Wilson, S. B. (2006). Procedural debiasing of primacy/anchoring effects in clinical-like judgments. *Journal of Clinical Psychology*, *51*, 841–853.

Parducci, A. (1965). Category judgment: A range-frequency model. *Psychological Review*, *72*, 407–418.

Parducci, A. (1968). The relativism of absolute judgment. *Scientific American*, *219*, 84–90.

Petrov, A. A., & Anderson, J. R. (2005). The dynamics of scaling: A memory-based anchor model of category rating and identification. *Psychological Review*, *112*, 383–416.

Stewart, N., Brown, G. D. A., & Chater, N. (2005). Absolute identification by relative judgment. *Psychological Review*, *112*, 881–911.

Wedell, D. H., Parducci, A., & Lane, M. (1990). Reducing the dependence of clinical judgment on the immediate context: Effects of number of categories and type of anchors. *Journal of Personality and Social Psychology*, *58*, 319–329.

